# Adaptively Growing Hierarchical Mixtures of Experts

Jürgen Fritsch, Michael Finke, Alex Waibel
{fritsch+,finkem,waibel}@cs.cmu.edu
Interactive Systems Laboratories
Carnegie Mellon University
Pittsburgh, PA 15213

## Abstract

We propose a novel approach to automatically growing and pruning Hierarchical Mixtures of Experts. The constructive algorithm proposed here enables large hierarchies consisting of several hundred experts to be trained effectively. We show that HME's trained by our automatic growing procedure yield better generalization performance than traditional static and balanced hierarchies. Evaluation of the algorithm is performed (1) on vowel classification and (2) within a hybrid version of the JANUS [9] speech recognition system using a subset of the Switchboard large-vocabulary speaker-independent continuous speech recognition database.

## INTRODUCTION

The Hierarchical Mixtures of Experts (HME) architecture [2,3,4] has proven useful for classification and regression tasks in small to medium sized applications with convergence times several orders of magnitude lower than comparable neural networks such as the multi-layer perceptron. The HME is best understood as a probabilistic decision tree, making use of soft splits of the input feature space at the internal nodes, to divide a given task into smaller, overlapping tasks that are solved by expert networks at the terminals of the tree. Training of the hierarchy is based on a generative model using the Expectation Maximisation (EM) [1,3] algorithm as a powerful and efficient tool for estimating the network parameters.

In [3], the architecture of the HME is considered pre-determined and remains fixed during training. This requires choice of structural parameters such as tree depth and branching factor in advance. As with other classification and regression techniques, it may be advantageous to have some sort of data-driven model-selection mechanism to (1) overcome false initialisations (2) speed-up training time and (3) adapt model size to task complexity for optimal generalization performance. In [11], a constructive algorithm for the HME is presented and evaluated on two small classification tasks: the two spirals and the 8-bit parity problems. However, this

algorithm requires the evaluation of the increase in the overall log-likelihood for all potential splits (all terminal nodes) in an existing tree for each generation. This method is computationally too expensive when applied to the large HME's necessary in tasks with several million training vectors, as in speech recognition, where we can not afford to train all potential splits to eventually determine the single best split and discard all others. We have developed an alternative approach to growing HME trees which allows the fast training of even large HME's, when combined with a path pruning technique. Our algorithm monitors the performance of the hierarchy in terms of scaled log-likelihoods, assigning penalties to the expert networks, to determine the expert that performs worst in its local partition. This expert will then be expanded into a new subtree consisting of a new gating network and several new expert networks.

## HIERARCHICAL MIXTURES OF EXPERTS

We restrict the presentation of the HME to the case of classification, although it was originally introduced in the context of regression. The architecture is a tree with gating networks at the non-terminal nodes and expert networks at the leaves. The gating networks receive the input vectors and divide the input space into a nested set of regions, that correspond to the leaves of the tree. The expert networks also receive the input vectors and produce estimates of the a-posteriori class probabilities which are then blended by the gating network outputs. All networks in the tree are linear, with a softmax non-linearity as their activation function. Such networks are known in statistics as multinomial logit models, a special case of Generalized Linear Models (GLIM) [5] in which the probabilistic component is the multinomial density. This allows for a probabilistic interpretation of the hierarchy in terms of a generative likelihood-based model. For each input vector $\mathbf{x}$, the outputs of the gating networks are interpreted as the input-dependent multinomial probabilities for the decisions about which child nodes are responsible for the generation of the actual target vector $\mathbf{y}$. After a sequence of these decisions, a particular expert network is chosen as the current classifier and computes multinomial probabilities for the output classes. The overall output of the hierarchy is

$$P(\mathbf{y}|\mathbf{x}, \Theta) = \sum_{i=1}^{N} g_i(\mathbf{x}, \mathbf{v}_i) \sum_{j=1}^{N} g_{j|i}(\mathbf{x}, \mathbf{v}_{ij}) P(\mathbf{y}|\mathbf{x}, \theta_{ij})$$

where the $g_i$ and $g_{j|i}$ are the outputs of the gating networks.

The HME is trained using the EM algorithm [1] (see [3] for the application of EM to the HME architecture). The E-step requires the computation of posterior node probabilities as expected values for the unknown decision indicators:

$$h_i = \frac{g_i \sum_j g_{j|i} P_{ij}(\mathbf{y})}{\sum_i g_i \sum_j g_{j|i} P_{ij}(\mathbf{y})} \qquad h_{j|i} = \frac{g_{j|i} P_{ij}(\mathbf{y})}{\sum_j g_{j|i} P_{ij}(\mathbf{y})}$$

The M-step then leads to the following independent maximum-likelihood equations

$$\theta_{ij} = \arg\max_{\theta_{ij}} \sum_t h_{ij}^{(t)} \log P_{ij}(\mathbf{y}^{(t)})$$

$$\mathbf{v}_i = \arg\max_{\mathbf{V}_i} \sum_t \sum_k h_k^{(t)} \log g_k^{(t)}$$

$$\mathbf{v}_{ij} = \arg\max_{\mathbf{V}_{ij}} \sum_t \sum_k h_k^{(t)} \sum_l h_{l|k}^{(t)} \log g_{l|k}^{(t)}$$

where the $\theta_{ij}$ are the parameters of the expert networks and the $v_i$ and $v_{ij}$ are the parameters of the gating networks. In the case of a multinomial logit model, $P_{ij}(\mathbf{y}) = y_c$, where $y_c$ is the output of the node associated with the correct class. The above maximum likelihood equations might be solved by gradient ascent, weighted least squares or Newton methods. In our implementation, we use a variant of Jordan & Jacobs' [3] least squares approach.

## GROWING MIXTURES

In order to grow an HME, we have to define an evaluation criterion to score the experts performance on the training data. This in turn will allow us to select and split the worst expert into a new subtree, providing additional parameters which can help to overcome the errors made by this expert. Viewing the HME as a probabilistic model of the observed data, we partition the input dependent likelihood using expert selection probabilities provided by the gating networks

$$
\begin{aligned}
l(\Theta; \mathcal{X}) &= \sum_t \log P(\mathbf{y}^{(t)}|\mathbf{x}^{(t)}, \Theta) = \sum_t \sum_k g_k \log P(\mathbf{y}^{(t)}|\mathbf{x}^{(t)}, \Theta) \\
&= \sum_k \sum_t \log[P(\mathbf{y}^{(t)}|\mathbf{x}^{(t)}, \Theta)]^{g_k} = \sum_k l_k(\Theta; \mathcal{X})
\end{aligned}
$$

where the $g_k$ are the products of the gating probabilities along the path from the root node to the $k$-th expert. $g_k$ is the probability that expert $k$ is responsible for generating the observed data (note, that the $g_k$ sum up to one). The expert-dependent scaled likelihoods $l_k(\Theta; \mathcal{X})$ can be used as a measure for the performance of an expert within its region of responsibility. We use this measure as the basis of our tree growing algorithm:

1. Initialize and train a simple HME consisting of only one gate and several experts.

2. Compute the expert-dependent scaled likelihoods $l_k(\Theta; \mathcal{X})$ for each expert in one additional pass through the training data.

3. Find the expert $k$ with minimum $l_k$ and expand the tree, replacing the expert by a new gate with random weights and new experts that copy the weights from the old expert with additional small random perturbations.

4. Train the architecture to a local minimum of the classification error using a cross-validation set.

5. Continue with step (2) until desired tree size is reached.

The number of tree growing phases may either be pre-determined, or based on difference in the likelihoods before and after splitting a node. In contrast to the growing algorithm in [11], our algorithm does not hypothesize all possible node splits, but determines the expansion node(s) directly, which is much faster, especially when dealing with large hierarchies. Furthermore, we implemented a path pruning technique similar to the one proposed in [11], which speeds up training and testing times significantly. During the recursive depth-first traversal of the tree (needed for forward evaluation, posterior probability computation and accumulation of node statistics) a path is pruned temporarily if the current node's probability of activation falls below a certain threshold. Additionally, we also prune subtrees permanently, if the sum of a node's activation probabilities over the whole training set falls below a certain threshold. This technique is consistent with the growing algorithm and also helps preventing instabilities and singularities in the parameter updates, since nodes that accumulate too little training information will not be considered for a parameter update because such nodes are automatically pruned by the algorithm.

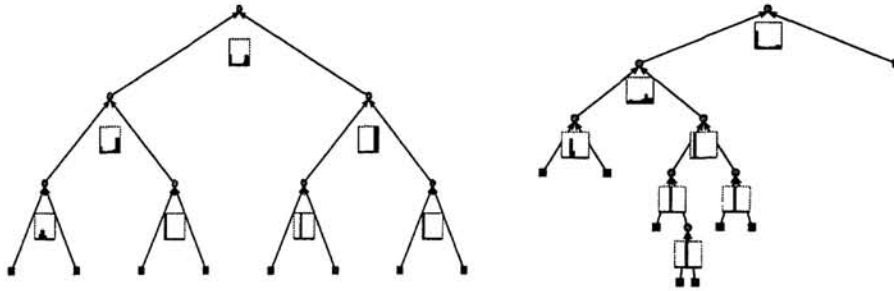

Figure 1: Histogram trees for a standard and a grown HME

## VOWEL CLASSIFICATION

In initial experiments, we investigated the usefulness of the proposed tree growing algorithm on Peterson and Barney's [6] vowel classification data that uses formant frequencies as features. We chose this data set since it is small, non-artificial and low-dimensional, which allows for visualization and understanding of the way the growing HME tree performs classification tasks.

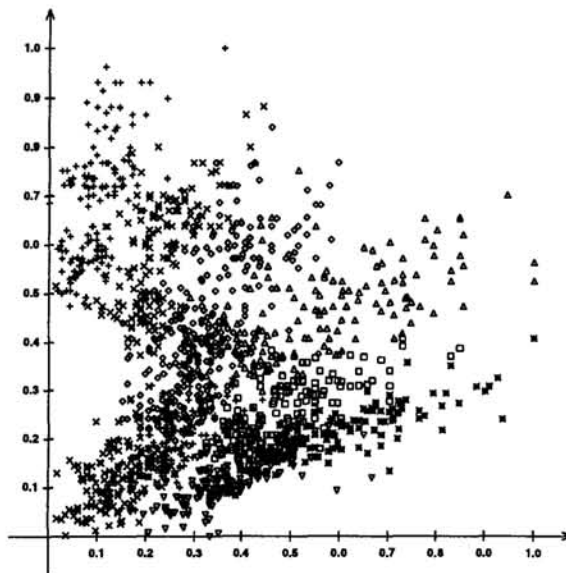

The vowel data set contains 1520 samples consisting of the formants F0, F1, F2 and F3 and a class label, indicating one of 10 different vowels. Experiments were carried out on the 4-dimensional feature space, however, in this paper graphical representations are restricted to the F1-F2 plane. The figure to the left shows the data set represented in this plane (The formant frequencies are normalized to the range [0,1]).

In the following experiments, we use binary branching HME's exclusively, but in general the growing algorithm poses no restrictions on the tree branching factor. We compare a standard, balanced HME of depth 3 with an HME that grows from a two expert tree to a tree with the same number of experts (eight) as the standard HME. The size of the standard HME was chosen based on a number of experiments with different sized HME's to find an optimal one. Fig. 1 shows the topology of the standard and the fully grown HME together with histograms of the gating probability distributions at the internal nodes.

Fig. 2 shows results on 4-dimensional feature vectors in terms of correct classification rate and log-likelihood. The growing HME achieved a slightly better (1.6% absolute) classification rate than the fixed HME. Note also, that the growing HME outperforms the fixed HME even before it reaches its full size. The growing HME was expanded every 4 iterations, which explains the bumpiness of the curves.

Fig. 3 shows the impact of path pruning during training on the final classification rate of the grown HME's. The pruning factor ranges from no pruning to full pruning (e.g. only the most likely path survives).

Fig. 4 shows how the gating networks partition the feature space. It contains plots

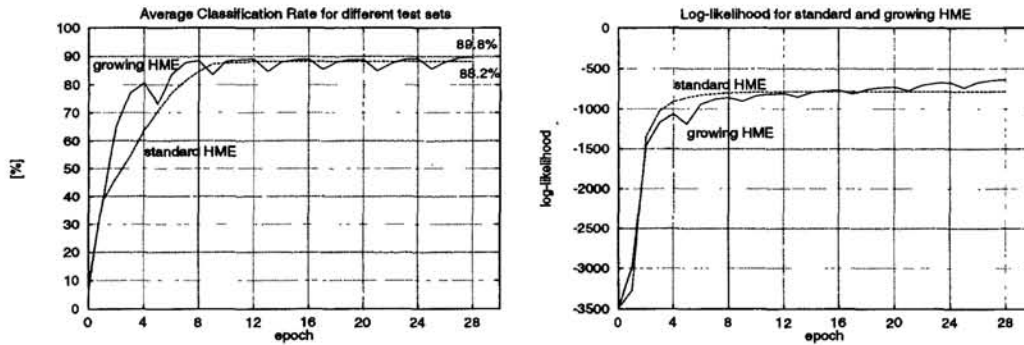

Figure 2: Classification rate and log-likelihood for standard and growing HME

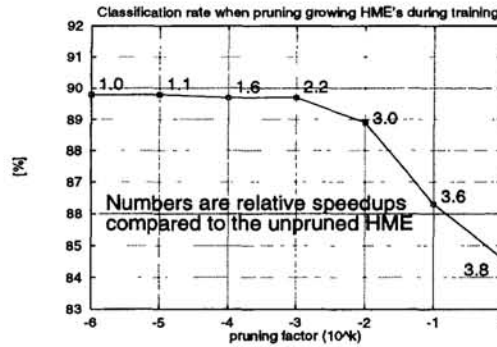

Figure 3: Impact of path pruning during training of growing HME's

of the activation regions of all 8 experts of the standard HME in the 2-dimensional range $[-0.1, 1.1]^2$. Activation probabilities (product of gating probabilities from root to expert) are colored in shades of gray from black to white. Fig. 5 shows the same kind of plot for all 8 experts of the grown HME. The plots in the upper right corner illustrate the class boundaries obtained by each HME.

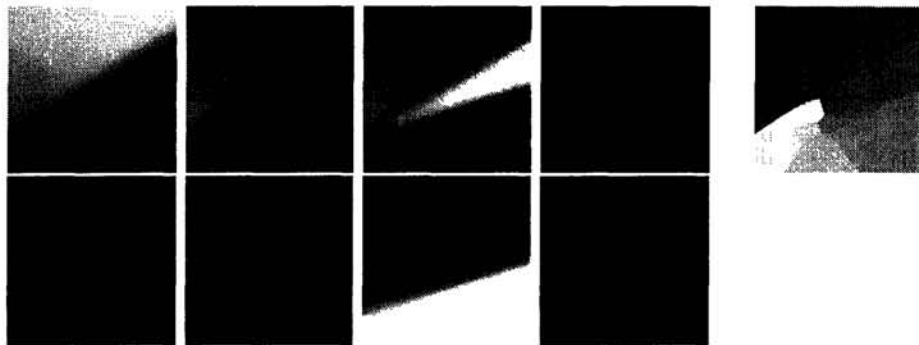

Figure 4: Expert activations for standard HME

Fig. 4 reveals a weakness of standard HME's: Gating networks at high levels in the tree can pinch off whole branches, rendering all the experts in the subtree useless. In our case, half of the experts of the standard HME do not contribute to the final decision at all (black boxes). The growing HME's are able to overcome this effect. All the experts of the grown HME (Fig. 5) have non-zero activation patterns and the overlap between experts is much higher in the growing case, which indicates a higher degree of cooperation among experts. This can also be seen in the histogram trees in Fig. 3, where gating networks in lower levels of the grown tree tend to

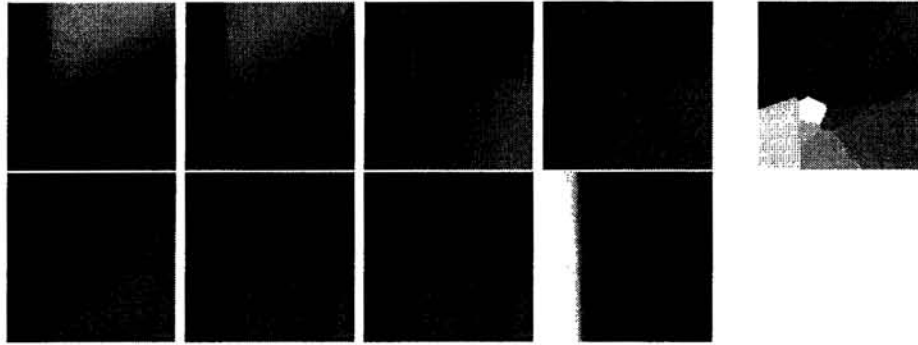

Figure 5: Expert activations for grown HME

average the experts outputs. The splits formed by the gating networks also have implications on the way class boundaries are formed by the HME. There are strong dependencies visible between the class boundaries and some of the experts activation regions.

## EXPERIMENTS ON SWITCHBOARD

We recently started experiments using standard and growing HME's as estimators of posterior phone probabilities in a hybrid version of the JANUS [9] speech recognizer. Following the work in [12], we use different HME's for each state of a phonetic HMM. The posteriors for 52 phonemes computed by the HME's are converted into scaled likelihoods by dividing by prior probabilities to account for the likelihood based training and decoding of HMM's. During training, targets for the HME's are generated by forced-alignment using a baseline mixture of Gaussian HMM system. We evaluate the system on the Switchboard spontaneous telephone speech corpus. Our best current mixture of Gaussians based context-dependent HMM system achieves a word accuracy of 61.4% on this task, which is among the best current systems [7]. We started by using phonetic context-independent (CI) HME's for 3-state HMM's. We restricted the training set to all dialogues involving speakers from one dialect region (New York City), since the whole training set contains over 140 hours of speech. Our aim here was, to reduce training time (the subset contains only about 5% of the data) to be able to compare different HME architectures.

| Context | ♯ HME | branching | ♯ experts | Word Acc. |
|---|---|---|---|---|
| CI | 3 | 4 | 64 | 33.8% |
| CI growing | 3 | 4 | 64 | 35.1% |
| CD/CI | 3x52 | 8/4 | 8/64 | 42.1% |
| CD/CI growing | 3x52 | 2/4 | 8/64 | 45.3% |

Figure 6: Preliminary results on Switchboard telephone data

To improve performance, we then build context-dependent (CD) models consisting of a separate HME for each biphone context and state. The CD HME's output is smoothed with the CI models based on prior context probabilities. Current work focuses on improving context modeling (e.g. larger contexts and decision tree based clustering).

Fig. 6 summarizes the results so far, showing consistently that growing HME's outperform equally sized standard HME's. The results are not directly comparable

with our best Gaussian mixture system, since we restricted context modeling to biphones and used only a small subset of the Switchboard database for training.

## CONCLUSIONS

In this paper, we presented a method for adaptively growing Hierarchical Mixtures of Experts. We showed, that the algorithm allows the HME to use the resources (experts) more efficiently than a standard pre-determined HME architecture. The tree growing algorithm leads to better classification performance compared to standard HME's with equal numbers of parameters. Using growing instead of fixed HME's as continuous density estimators in a hybrid speech recognition system also improves performance.

## References

[1] Dempster, A.P., Laird, N.M. & Rubin, D.B. (1977) Maximum likelihood from incomplete data via the EM algorithm. *J.R. Statist. Soc. B 39* , 1-38.

[2] Jacobs, R. A., Jordan, M. I., Nowlan, S. J., & Hinton, G. E. (1991) Adaptive mixtures of local experts. In *Neural Computation 3*, pp. 79-87, MIT press.

[3] Jordan, M.I. & Jacobs R.A. (1994) Hierarchical Mixtures of Experts and the EM Algorithm. In *Neural Computation 6*, pp. 181-214. MIT press.

[4] Jordan, M.I. & Jacobs, R.A. (1992) Hierarchies of adaptive experts. In *Advances in Neural Information Processing Systems 4*, J. Moody, S. Hanson, and R. Lippmann, eds., pp. 985-993. Morgan Kaufmann, San Mateo, CA.

[5] McCullagh, P. & Nelder, J.A. (1983) *Generalized Linear Models.* Chapman and Hall, London.

[6] Peterson, G. E. & Barney, H. L. (1952) Control measurements used in a study of the vowels. *Journal of the Acoustical Society of America 24*, 175-184.

[7] Proceedings of LVCSR Hub 5 workshop, Apr. 29 - May 1 (1996) MITAGS, Linthicum Heights, Maryland.

[8] Syrdal, A. K. & Gopal, H. S. (1986) A perceptual model of vowel recognition based on the auditory representation of American English vowels. *Journal of the Acoustical Society of America, 79* (4):1086-1100.

[9] Zeppenfeld T., Finke M., Ries K., Westphal M. & Waibel A. (1997) Recognition of Conversational Telephone Speech using the Janus Speech Engine. *Proceedings of ICASSP 97, Muenchen, Germany*

[10] Waterhouse, S.R., Robinson, A.J. (1994) Classification using Hierarchical Mixtures of Experts. In *Proc. 1994 IEEE Workshop on Neural Networks for Signal Processing IV*, pp. 177-186.

[11] Waterhouse, S.R., Robinson, A.J. (1995) Constructive Algorithms for Hierarchical Mixtures of Experts. In *Advances in Neural Information Processing Systems 8*.

[12] Zhao, Y., Schwartz, R., Sroka, J. & Makhoul, J. (1995) Hierarchical Mixtures of Experts Methodology Applied to Continuous Speech Recognition. In *ICASSP 1995*, volume 5, pp. 3443-6, May 1995.
